# A Simple yet Universal Framework for Depth Completion

**Jin-Hwi Park**
AI Graduate School
GIST
jinhwipark@gm.gist.ac.kr

**Hae-Gon Jeon**
AI Graduate School
GIST
haegonj@gist.ac.kr

## Abstract

Consistent depth estimation across diverse scenes and sensors is a crucial challenge in computer vision, especially when deploying machine learning models in the real world. Traditional methods depend heavily on extensive pixel-wise labeled data, which is costly and labor-intensive to acquire, and frequently have difficulty in scale issues on various depth sensors. In response, we define **Uni**versal **D**epth **C**ompletion (UniDC) problem. We also present a baseline architecture, a simple yet effective approach tailored to estimate scene depth across a wide range of sensors and environments using minimal labeled data. Our approach addresses two primary challenges: *generalizable knowledge* of unseen scene configurations and *strong adaptation* to arbitrary depth sensors with various specifications. To enhance versatility in the wild, we utilize a foundation model for monocular depth estimation that provides a comprehensive understanding of 3D structures in scenes. Additionally, for fast adaptation to off-the-shelf sensors, we generate a pixel-wise affinity map based on the knowledge from the foundation model. We then adjust depth information from arbitrary sensors to the monocular depth along with the constructed affinity. Furthermore, to boost up both the adaptability and generality, we embed the learned features into hyperbolic space, which builds implicit hierarchical structures of 3D data from fewer examples. Extensive experiments demonstrate the proposed method's superior generalization capabilities for UniDC problem over state-of-the-art depth completion. Source code is publicly available at https://github.com/JinhwiPark/UniDC.

## 1 Introduction

Acquiring accurate and dense depth maps is crucial for various computer vision tasks such as scene understanding [1, 2, 3, 4], 3D reconstruction [5, 6, 7, 8], and autonomous driving [9, 10, 11]. Traditional methods like dense stereo matching [12, 13, 14] often face challenges of handling occlusion and varying lighting conditions between viewpoints. Additionally, depth maps obtained from active depth sensors [15, 16] like LiDAR and Time-of-Flight cameras typically exhibit low resolutions. As a solution to the above problems, depth completion has been widely studied. The goal of depth completion is to obtain a depth map from a pair of an image and a low-resolution depth map (often sparse depth map) taken by active sensors. The depth completion aims to convert a sparse depth map into a dense depth prediction by propagating it with an image-based affinity map.

Recent advances in learning-based depth perceptions have markedly improved the performance in this domain; however, most approaches are still tailored to specific settings and struggle to generalize to new environments or sensor types. While generalizable knowledge can be achieved by training huge models with large-scale and diverse datasets, acquiring accurate and dense depth information as ground-truth data is prohibitively expensive and time-consuming, which makes such a generalization model for metric scale 3D depth prediction infeasible in practice. Moreover, there exist numerous types of active depth sensors and complex scenarios in the real world. Unfortunately, only two

benchmark datasets (*e.g.*, KITTI [17] and NYU dataset [18]) are predominantly utilized in relevant research fields. Considering the accessibility of various industrial scenarios and the extremely high annotation cost, it is desirable to explore a few-shot learning approach capable of universal depth prediction for both arbitrary sensors and environments.

In response to the growing needs of both industry and the research community, in this work, we define a new problem, called **Uni**versal **D**epth **C**ompletion (UniDC), and present a baseline architecture and its advanced version. Our key insight of the baseline model for UniDC is to utilize pre-trained knowledge from a foundation model for monocular depth estimation, which provides depth-aware information enriched with high-resolution contextual information. Previous works typically exploit entangled representations of an image and corresponding depth data by concatenating them in an input layer, which reduces the generality of the foundation model. A contemporary work [19] proposes a sensor-agnostic depth completion with a depth prompting module, which mitigates the sensor bias problem by disentangling image and depth modalities. Since the depth representation is optimized with respect to a specific scene environment, it has limitations in out-of-domain situations, such as the environmental transition from indoor to outdoor, and vice versa.

To resolve this limitation, we design a simple baseline architecture using the foundation model. By excluding the training procedure for a new encoder to represent depth data, we achieve a high generality of the model across various sensors regardless of scene configurations. The proposed architecture consists of three sequential steps: (1) extraction of depth-aware features from the foundation model; (2) sparse-to-dense conversion based on the depth-aware information; (3) refinement of the converted depth with a pixel-wise affinity map constructed based on high-resolution details of the input image. For more details, the sparse-to-dense conversion aggregates adjacent depth values based on the high-resolution pixel-wise features from the foundation model. In the depth refinement process, we adopt a spatial propagation module with a multi-kernel affinity map.

We next boost up the baseline architecture by taking advantage of hyperbolic embedding. As stated in [20, 21], the natural capacity of hyperbolic spaces encourages capturing the implicit hierarchical structure of 3D data. In particular, this capability alleviates bleeding errors in the spatial propagation process [22]. To ensure adaptability and generality, we also design a multi-curvature approach for producing multiple affinity maps in the refinement stage. The effectiveness of our models is demonstrated across a variety of scenarios and datasets, confirming its superior generalization and robustness in different sensor setups and scene configurations. We also conduct extensive experiments and analyses to validate the efficacy of the proposed model.

## 2  Related Works

***Depth Completion.*** Image-guided depth completion aims to predict dense depth maps from an RGB image and its synchronized sparse depth acquired by depth sensors. A work in [23] introduces a deep regression model that significantly enhances prediction accuracy over the existing monocular depth estimation method [24], which utilizes only RGB image as input. However, depth maps from the direct regression method often suffer from blurry artifacts and distortions at object boundaries [25]. To address these issues, several works have introduced spatial propagation networks (SPNs) [25, 26, 27, 28, 29] as refinement modules. SPNs iteratively update the output of direct-regression methods by aggregating neighboring pixels over a reference pixel. Nonetheless, these models are typically tailored for specific depth sensors, such as the 64-Line Velodyne LiDAR [30] in KITTI outdoor dataset [17] and Kinect [31] for NYUv2 indoor dataset [18].

To alleviate this limited usage of SPNs, several studies have explored sensor-/domain-agnostic depth completion. SpAgNet [32] develops a model agnostic to the sparsity of depth points by incorporating sparse depth representations into a depth decoder. Another work [33] takes the use of both sparse metric depth and data-driven priors from a monocular depth prediction network for domain-agnostic depth completion. DepthPrompting [19] solves sensor bias problems with a prompt engineering. Despite these efforts, they still face challenges with a cross-domain generalization [19] and an issue on a limitation of sensors' scan ranges, which causes an overfitting problem [32, 33].

***Usage of Foundation Model in Downstream Task.*** Foundation models, designed for various downstream tasks, have revolutionized both natural language processing and computer vision fields. In particular, in the computer vision field, these foundation models excel in high-level visual perception tasks such as image recognition [34, 35, 36] and image captioning [35, 37, 36]. Those

vision foundation models provide benefits for strong adaptation to various tasks via tuning methods [38, 39, 40, 41] and feature adaptation methods [42, 43, 44]. In low-level tasks like depth computation, several works [45, 46, 47] create diverse datasets for zero-shot generalization capabilities, while others [48, 49, 50] fine-tune the text-to-image model [51] to utilize diffusion priors for better generalization which guides them to keep geometric details.

***Hyperbolic Geometry for Visual Data.*** Hyperbolic embedding for efficient learning-based approaches [52, 53] has gained interest. The Hyperbolic embedding has validated its ability to effectively represent complex data as hierarchical structures in low-dimensional spaces, offering a distinct advantage over Euclidean embeddings. This unique capability promotes the design of hyperbolic neural networks, and is applicable for a range of applications such as hierarchical recognition [54, 55, 56], retrieval [57, 58, 59], dealing with uncertainty [60, 61, 62], and generative learning on scarce data [63, 64, 65, 66]. Especially, hyperbolic methods have been shown to be effective in addressing low-shot visual problems [67, 68, 69, 60, 70], modeling complex 3D data [20, 21] and measuring pixel-wise similarity [22]. In this work, we devise the hyperbolic version of the proposed architecture to make both the generalizable power and understanding 3D depth data better.

# 3  Baseline Architecture

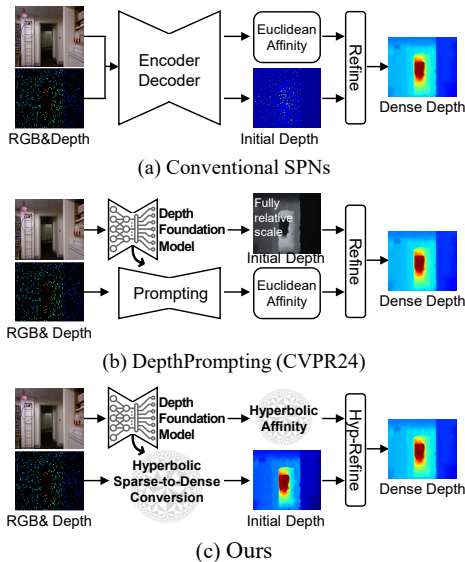

(a) Conventional SPNs

(b) DepthPrompting (CVPR24)

(c) Ours

Figure 1: Illustrations of conventional SPN, sensor agnostic model [19] and ours. Our approach uses hyperbolic-based depth completion in three stages: generating an initial depth, constructing a pixel-wise affinity, and refining the depth based on the affinity.

We present a simple yet effective architecture to achieve a generalizable depth completion model for unseen environments with only minimal data. Firstly, in Sec.3.1, we explain the rationale for adopting a monocular depth foundation model to simultaneously achieve sensor-/domain-agnostic depth completion. We then propose a baseline model architecture for UniDC, which integrates the pre-trained foundation model with both the depth propagation and refinement process in Sec.3.2.

## 3.1  Rationale: Foundation Model Usage in UniDC

***Difficulties to generalize depth completion.*** The two major obstacles to sensor-/domain-agnostic depth completion are the high cost of dense depth data acquisition and the scale variance across different sensors. First, capturing dense depth data on a metric scale is expensive. For example, Velodyne 64-line LIDAR, used in the KITTI dataset, provides high-quality depth information but has less than 6% density relative to the number of pixels in its synchronized image. Second, sensors have their own scanning ranges, hindering the development of a universal solution. As shown in Fig. 1-(a,b), the previous frameworks learn the joint representation of image and depth, and the depth prompting module, respectively. However, the trained encoder is vulnerable to handling different sensors due to a bias towards specific scanning ranges.

***Usage of depth-aware knowledge from depth foundation model.*** Although the depth foundation model produces relative depth maps, we can measure pixel-wise similarity using them. For example, we are able to distinguish between foreground and background regions only with the relative depth maps and account for depth boundaries between objects in scenes. Therefore, based on this depth-aware information, it will be the most probable solution that propagates a given sparse metric depth into the remaining pixels in an input image space without any additional learning for the depth.

***Revisiting how to use SPN.*** SPN [71] constitutes a core component in most state-of-the-art (SoTA) depth completion and is typically invoked as a final refinement step. The SPN refinement module takes initial depth and pixel-wise affinity as input and yields refined dense depth by iteratively updating its output. During training, the previous methods (Fig.1-(a,b)) jointly optimize the pixel-wise affinity and initial depth. However, the joint optimization scheme hinders the fast adaptation to new environments because learned weights are asked to have both domain- and depth-specific features. Furthermore,

DepthPrompting (Fig.1-(b)), which employs a depth foundation model for a relative-scale depth map as initial depth of SPN, struggles to adapt to new environments with a limited data. We want to eliminate the possibility of degeneracy, so we devise a sparse-to-dense conversion with a foundation model to make an initial dense depth. In Fig.1, different from the coarse initial depth seen in traditional SPNs, our method provides promising results even before the SPN refinement step.

## 3.2 Architecture Design

Considering the facts discussed in Sec.3.1, we devise an effective baseline architecture. We first utilize pre-trained knowledge from a foundation model tailored for monocular depth estimation, which provides pixel-wise relative distances (a.k.a. relative scene depth) from a camera along with high-resolution contextual information. Thanks to the knowledge, our baseline architecture becomes simpler due to no need for an additional encoder to represent depth data from arbitrary sensors. Our model operates in three stages: ① extraction of the relative depth-aware features from the foundation model, ② propagation of spare depth from arbitrary sensors based on the depth-aware features, and ③ refinement of it with a pixel-wise affinity map constructed from the depth-aware features. This scheme not only simplifies the architectural complexity, but also enhances the adaptability and performance across diverse sensing scenarios. The overall algorithm scheme is summarized in Alg.1.

***Tuning strategy for foundation model.*** Given a single image $I \in \mathbb{R}^{3 \times H \times W}$, the pre-trained depth model $f_{\mathcal{F}}$ outputs multi-scale intermediate features $E$ and relative depth $D_{\text{relative}}$ as below:

$$E, D_{\text{relative}} = f_{\mathcal{F}}(I, \Theta_{f_{\mathcal{F}}}), \tag{1}$$

where $\Theta_{f_{\mathcal{F}}}$ denotes parameters of the foundation model.

Since the foundation model is trained to estimate relative depth from single images, they inherently face limitations when handling metric scale depths. To reduce the modality discrepancy, our approach involves an integration of an additional loss term to refine the foundation model by minimizing the difference between $D_{\text{relative}}$ and its Ground Truth (GT) depth $D_{gt}$ for valid pixels $v \in V$. Let $\delta_v = \log D_{\text{relative}}(v) - \log D_{gt}(v)$, the loss $L_{\text{scale-invariant}}$ is defined as below:

$$L_{\text{scale-invariant}}(D_{\text{relative}}, D_{gt}) = \frac{1}{|V|} \sum_{v \in V} (\delta_v)^2 - \frac{\lambda}{|V|^2} \left( \sum_{v \in V} \delta_v \right)^2, \tag{2}$$

where we set $\lambda = 0.85$ in all experiments as in [24]. We also implement a bias tuning [38, 39], shown to be more effective for dense prediction tasks than other tuning protocols [72, 38]. The bias tuning updates the bias terms while keeping the rest of the backbone parameters unchanged, thus preserving the high-resolution details and contextual information. These strategic modifications significantly enhance the capability of the foundation model for estimating metric scale depth.

# 4 Advanced Architecture with Hyperbolic Geometry

We also present an advanced version of the baseline architecture that grafts hyperbolic geometry onto the depth foundation model, known for its effectiveness in low-shot problems [67, 68, 69, 60, 70]. We first generate depth-aware features by merging the multi-scale intermediate features $E$ derived from the foundation model and by embedding them into hyperbolic space with geometry-aware curvature (Sec.4.1). Using the depth-aware features alongside sparse sensor data, we develop a hyperbolic propagation inspired by a traditional bilateral filter mechanism, which yields an initial dense depth at a metric scale (Sec.4.2). We lastly introduce a process for generating multi-curvature hyperbolic space for high-fidelity pixel relations and refinement of the initial depth (Sec.4.3).

## 4.1 Multi-scale Feature Fusion & Hyperbolic Curvature Generation

The intermediate features from the foundation model $E_l \in E$, where $l = 0, \ldots, L-1$, correspond to scales factors $1/2, \ldots, 1/2^L$ of the original resolution of input images. We aim to synergistically fuse the multi-scale information to learn comprehensive, context-aware features that facilitate depth propagation at a metric scale. We upsample the coarser feature map $E_l^M (E_0^M = E_0)$ using convolution layers, and then aggregate $E_l^M$ with finer feature map $E_{l+1}$ to obtain better visual contextual features

---

**Algorithm 1** Implementation of Hyperbolic Universal Depth Completion

---

**Require:** Given a single image $I \in \mathbb{R}^{3 \times H \times W}$, depth foundation model $f_{\mathcal{F}}$ and the corresponding parameter $\Theta_{f_{\mathcal{F}}}$, multi-scale feature aggregation blocks $f_l^{fusion}$, number of multi-scale feature $L$, curvature generation blocks $\mathcal{C}$, set of neighboring pixel coordinate $N(i)$, kernel function $\mathcal{P}$, and multi-kernel affinity map $A_k$.

1: **procedure**
2:     $E = f_{\mathcal{F}}(I, \Theta_{f_{\mathcal{F}}})$                                                        $\triangleright$ Multi-scale Features Extraction (Eq.1)
3:     **[Stage-①] Multi-scale Feature Fusion & Hyperbolic Curvature Generation**
4:     **for** $E_l$ in $E$ **do**     $(l = 0, \ldots, L-1)$
5:         $E_{l+1}^M = f_l^{fusion}(E_l^M, E_{l+1})$   $(E_0^M = E_0)$                             $\triangleright$ Feature Fusion (Eq.3)
6:     **end for**
7:     $\kappa = \mathcal{C}(E_L^M)$                                                $\triangleright$ Curvature Generation (Eq.6)
8:     **for** $E_{L,i}^M, E_{L,j}^M \in N(i)$ in $E_L^M$ **do**
9:         **[Stage-②] Sparse-to-Dense Conversion based on Hyperbolic Features**
10:         $H_i = \exp_0^\kappa(E_{L,i}^M), \quad H_j = \exp_0^\kappa(E_{L,j}^M)$            $\triangleright$ Hyperbolic Embedding (Eq.5)
11:         $w_{ij} = \mathcal{P}(Dist_{hyp}(H_i, H_j), Dist_{euc}(E_{L,i}^M, E_{L,j}^M))$     $\triangleright$ Hyperbolic Kernel (Eq.8)
12:         $D_i^{init} = \sum_j w_{ij} S_j$                                         $\triangleright$ Init Depth (Eq.8)
13:         **[Stage-③] Depth Refinement in Multi-curvature Hyperbolic Space**
14:         $\kappa_k = \mathcal{C}_k(E_L^M)$                                  $\triangleright$ Multi-curvature Generation (Eq.11)
15:         $A_k^{hyp} = HCL(E_{L,i}^M, \kappa_k)$                        $\triangleright$ Hyperbolic Affinity (Eq.11)
16:         $D_{i,k}^{t+1} = A_{i,k}^{hyp} \odot D_i^0 + \sum_{j \in \mathcal{N}_k(i)} A_{j,k}^{hyp} \odot D_{j,k}^t$     $\triangleright$ Hyperbolic Depth Refinement (Eq.9)
17:         $\hat{D}_i^{t+1} = \sum_{k \in \mathcal{K}} \sigma_{i,k} D_{i,k}^{t+1}$                              $\triangleright$ Final Depth (Eq.9)
18:     **end for**
19: **end procedure**

---

$E_{l+1}^M$. This fusion process is described below:

$$E_{l+1}^M = f_l^{fusion}(E_l^M, E_{l+1}), \tag{3}$$

where $f_l^{fusion}$ indicates multi-scale feature aggregation blocks consisting of 2D transposed convolution layers with a skip connection.

***Hyperbolic embedding.*** To ensure strong adaption to both new environments and any type of sensors, we adopt hyperbolic geometry which enables to capture the inherent hierarchical structures of 3D data [20, 21]. To embed the Euclidean features into hyperbolic space and vice versa, one first needs to define a bijective mapping from $\mathbb{R}^n$ to $\mathbb{D}_\kappa^n$. The exponential and the logarithmic mapping are used as bijective functions that have appealing forms at an origin, namely for $\mathbf{x} \in \mathbb{R}^\mathbf{n}$ and $\mathbf{u} \in \mathbb{D}_\kappa^\mathbf{n}$:

$$\exp_0^\kappa(\mathbf{x}) = \tanh(\sqrt{\kappa}\|\mathbf{x}\|/\mathbf{2})\frac{\mathbf{x}}{\sqrt{\kappa}\|\mathbf{x}\|} \quad \text{and} \quad \log_\mathbf{0}^\kappa(\mathbf{u}) = \tanh^{-\mathbf{1}}(\sqrt{\kappa}\|\mathbf{u}\|)\frac{\mathbf{u}}{\sqrt{\kappa}\|\mathbf{u}\|}. \tag{4}$$

Using hyperbolic geometry for pixel-wise relationships, especially spatial propagation, is demonstrated in [22] by improving the discriminative power with minimal supervision. Following [22], we embed the mixed feature $E_L^M$ into hyperbolic space using Eq.4 as below:

$$H_i = \exp_0^\kappa(E_{L,i}^M), \tag{5}$$

where $i$ is an index of spatial coordinates in the image domain, and $\kappa$ is the hyperbolic curvature.

***Hyperbolic curvature generation.*** Using an appropriate curvature value is an important factor in projecting Euclidean features into hyperbolic space well, which is closely related to the construction of the hierarchy structures. Previous methods mainly use a fixed geometric structure regardless of data types and scene configurations by merely adjusting $\kappa$ as a hyperparameter [70, 73, 74, 21]. In our problem definition, according to types of sensors and scene configurations, diverse data measurements and geometrical structures are observed, respectively. That's, our key observation is that a fixed and predetermined curvature may not be universally suitable.

We thus propose a curvature generation that learns a geometry-aware curved embedding space to adaptively match it to new environments and sensors. The curvature generator $\mathcal{C}$ is composed of a convolution layer, a multi-layer perceptron (MLP) layer, and a global mean-pooling over spatial dimensions, which yields scene-dependent curvatures based on the fused feature $E_L^M$ as below:

$$\kappa = \mathcal{C}(E_L^M). \tag{6}$$

## 4.2 Sparse-to-Dense Conversion based on Hyperbolic Features

With both the high-resolution pixel-wise features from the foundation model and the sparse depth data from arbitrary sensors, we perform a sparse-to-dense conversion to obtain an initial dense depth map. Inspired by [75], we design an initial propagation process based on a bilateral filtering mechanism [76], which is renowned for its edge-preserving ability by incorporating both radiometric differences and spatial distances into the bilateral weight. Considering a pixel $x_i$ and the corresponding neighborhood pixel $x_j$, the bilateral kernel filter $w_{ij}$ can be simply defined as:

$$w_{ij} = f_r(x_j, x_i) g_s(x_j - x_i), \tag{7}$$

where $f_r$ is a range kernel for radiometric differences. $g_s$ is a spatial kernel for physical separations in observed scenes and is developed in Euclidean space by calculating the distance between 3D points. For the range kernel $f_r$, we need to design its hyperbolic version. Here, we utilize the hyperbolic feature $H$ provided as input from Eq.5. With $f_r$ and $g_s$, we can compute the initial dense depth as:

$$D_i^{init} = \sum_j w_{ij} S_j \quad \text{s.t.} \quad w_{ij} = \mathcal{P}(Dist_{hyp}(H_i, H_j), Dist_{euc}(E_{L,i}^M, E_{L,j}^M)), \tag{8}$$

where $Dist_{hyp}$ is the hyperbolic function consisting of hyperbolic MLP, and $Dist_{euc}$ is the Euclidean distance in the 3-dimension coordinate. $N(i)$ means the neighborhood sparse depth of the pixel $i$, and $S_j$ is the corresponding depth from a sensor. $\mathcal{P}$ indicates the learnable MLP layer to compute a coefficient for each sparse depth of the neighborhood $S_j$. Through the combination of the distance functions in Eq.8, we effectively take advantage of both hyperbolic and Euclidean geometries to produce more accurate and robust depth maps.

## 4.3 Depth Refinement in Multi-curvature Hyperbolic Space

***Depth refinement.*** To refine the initial depth in Eq.8, we employ a convolutional spatial propagation scheme, CSPN++ [26]. This refinement process leverages a predefined depth map $D_i$, augmented by a sparse valid depth map $S$, and a multi-kernel affinity map with three different kernel sizes $\mathcal{K}=\{3, 5, 7\}$. The use of a multi-kernel approach enables the model to capture a diverse range of features from the input data, thus achieving detailed and comprehensive depth estimations. The propagation process for a kernel size $k \in \mathcal{K}$ at step $t$ to yield a dense map $\hat{D}$ is formulated as:

$$\hat{D}_i^{t+1} = \sum_{k \in \mathcal{K}} \sigma_{i,k} D_{i,k}^{t+1} \quad \text{s.t.} \quad D_{i,k}^{t+1} = A_{i,k} \odot D_i^0 + \sum_{j \in \mathcal{N}_k(i)} A_{j,k} \odot D_{j,k}^t, \tag{9}$$

where $D^t$ is the depth map at each propagation step $t$. $D^0$ and $A$ are an initial depth for $t = 0$ and its affinity map, respectively. $\odot$ is an element-wise product, and $j \in \mathcal{N}_k(i)$ denotes a set of neighboring pixels around pixel $i$ within a $k \times k$ window. $\sigma$ is a confidence map computed from $E_L^M$ in Sec.4.1.

***Hyperbolic convolution layer (HCL).*** We design the multi-kernel affinity map $A_k$ in hyperbolic space with the proposed curvature generation module described in Sec.4.1. To do this, we formulate HCL with hyperbolic feature vector $\mathbf{h}$ for a 2-dimensional image domain:

$$HCL(\mathbf{h}, \kappa) := \mathbf{W} \otimes_\kappa \mathcal{T}_{(i,j) \in \Omega}^\beta(\mathbf{h}) \oplus_\kappa \mathbf{b}, \tag{10}$$

where $\mathbf{W} \in \mathbb{R}^{C_{out} \times C_{in} \times \gamma \times \gamma}$ is a convolution weight matrix whose kernel size is $\gamma$, and $\mathbf{b}$ is a bias term. $\Omega = \{(i, j) \in \mathbb{Z}^2 \mid (-\gamma', -\gamma'), ..., (\gamma', \gamma'), \gamma' = \lfloor \frac{\gamma}{2} \rfloor\}$ is a set of signed distances from a center of the convolution kernel to others in $\mathbf{W}$. $\otimes$, $\oplus$ and $\mathcal{T}^\beta$ are hyperbolic multiplication, addition, and concatenation, respectively, whose details are in Appendix A.1. Note that the hyperbolic MLP (Eq.8) is designed with $\gamma = 1$.

***Mutli-curvature affinity generation.*** By dynamically adjusting the hyperbolic curvature $\kappa$ for each affinity map, our approach tailors the geometrical representation to better fit the specific depth structure of each scene. We first determine the hyperbolic curvature $\kappa$ with Eq.6 and then compute affinity map $A_k$ using a hyperbolic convolution operation equipped with a kernel of size $k$, chosen to match the receptive field of the corresponding kernel of the affinity map $A_k$. This alignment optimizes the local receptive fields across the depth map, enabling a more precise aggregation of context and texture information from neighboring pixels. We can calculate the hyperbolic affinity map $A_k^{hyp}$ based on the generated curvature $\kappa_k$ from the curvature generation blocks $\mathcal{C}_k$ as below:

$$A_k^{hyp} = HCL(E_{L,i}^M, \kappa_k) \quad \text{s.t.} \quad \kappa_k = \mathcal{C}_k(E_L^M). \tag{11}$$

We can achieve the refined dense depth based on the generated hyperbolic affinity maps $A_k^{hyp}$ by incorporating it into Eq.9. In particular, the employment of hyperbolic space is beneficial for depth perception by implicitly building hierarchical structures [22], whose roots come from sparse points of an input depth in this work. The hyperbolic space is also advantageous in regions where photometric distances between foreground

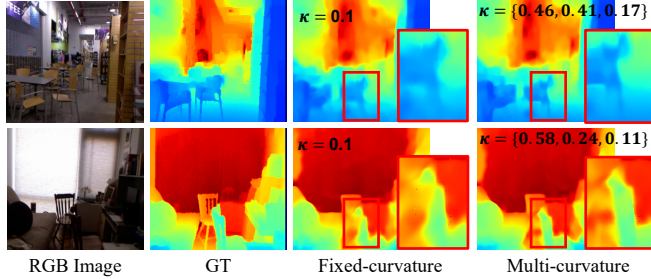

RGB Image　　GT　　Fixed-curvature　　Multi-curvature

Figure 2: Depth propagation results according to fixed and multi-curvature values.

and background pixels are marginal (see Fig.2). The detailed analysis of multi-curvature hyperbolic affinity is described in Sec. 5.3.

## 5    Experiment and Analysis

In this section, we evaluate the performance of our proposed method for UniDC, focusing on its adaptability using minimal labeled data. Firstly, we outline an overview of the experimental setup (Sec.5.1). Subsequent comparisons with various SoTA methods are then presented using standard benchmark datasets (Sec.5.2). Furthermore, we conduct an ablation study to clarify the impact of each component in our methodology (Sec.5.3). In Appendix.A.2, we introduce details of the training procedure, datasets, and evaluation metrics in this work. Additional experiments, including full dataset training benchmarks, hyperbolic space affinity calculations, an ablation study on foundation models, and varying-density performance, are included in the Appendix A.3.

### 5.1    Implementation Details

***Loss functions.*** We train our method in a supervised manner with a linear combination of two loss terms: scale-invariant loss [77] $L_{\text{scale-invariant}}$ (Eg.2) for bridging the gap between relative and metric scale depths, and a composite loss $L_{\text{L1L2}}$ based on $L_1$ and $L_2$ distances for inferring the final dense depth map. In total, our framework is optimized by minimizing the final loss $\mathcal{L}$ as below:

$$\mathcal{L} = L_{\text{L1L2}}(\hat{D}, D_{gt}) + \mu L_{\text{scale-invariant}}(D_{\text{relative}}, D_{gt}),$$
$$\text{s.t.}\quad L_{\text{L1L2}}(\hat{D}, D_{gt}) = \frac{1}{|V|} \sum_{i \in V} \left( \left| \hat{D}_i - D_{gt,i} \right| + \left| \hat{D}_i - D_{gt,i} \right|^2 \right). \tag{12}$$

where $\mu$ is a balance term and is empirically set to 0.1.

***Evaluation protocols.*** For fair evaluations, we select a diverse array of SoTA depth from sparse measurements. These include a sensor-agnostic model, DepthPrompting [19] and series of SPNs such as S2D [23], CSPN [25], NLSPN [78], DySPN [27], CostDCNet [79], CompletionFormer [80], and BPNet [75]. We assess depth quality using common quantitative metrics: root mean square error (RMSE, in meters), mean absolute error (MAE, in meters), and inlier ratio (DELTA1, where $\delta < 1.25$). We employ the widely-used depth completion datasets: NYU [18] and KITTI DC [81], setting up a minimal training dataset for few-shot scenarios. Note that we use their official test sets for all the comparison methods.

We implement the few-shot scenarios with and without dense depth supervision. Our experimental setup includes conducting 1-shot, 10-shot, and 100-shot learning by randomly sampling within the official training split. Additionally, we perform 1-sequence training by randomly selecting one sequence from the training set. To ensure the reliability in our experiments, we randomly select 10 sequences, and report averaged results.

### 5.2    Experiment

***Few-shot learning with dense GT.*** Both Tab.1 and Tab.3 show that existing methods face significant challenges when taking input depths from new sensors with minimal labeled data, whose examples are displayed in Fig.3 and Fig.4, respectively. In the 1-shot scenarios, where a model is optimized using

Table 1: Quantitative results on NYUv2.

| | 1-Shot | | | 10-Shot | | | 100-Shot | | | 1-Sequence Training | | |
|---|---|---|---|---|---|---|---|---|---|---|---|---|
| | RMSE | MAE | DELTA1 | RMSE | MAE | DELTA1 | RMSE | MAE | DELTA1 | RMSE | MAE | DELTA1 |
| CSPN [25] | 1.4827 | 1.2058 | 0.3455 | 0.3166 | 0.1961 | 0.7106 | 0.2854 | 0.1307 | 0.9748 | 0.3166 | 0.1961 | 0.7106 |
| NLSPN [78] | 1.9358 | 1.6132 | 0.2229 | 1.5995 | 0.8261 | 0.5040 | 0.5501 | 0.4150 | 0.7985 | 0.8881 | 0.6421 | 0.6809 |
| DySPN [27] | 1.5474 | 1.2851 | 0.3149 | 0.4102 | 0.2817 | 0.8595 | 0.2674 | 0.1706 | 0.9341 | 0.2584 | 0.1320 | 0.9615 |
| CompletionFormer [80] | 1.8218 | 1.5539 | 0.2408 | 1.1583 | 1.0162 | 0.3079 | 0.9914 | 0.8164 | 0.4379 | 0.6779 | 0.5356 | 0.7476 |
| CostDCNet [79] | 1.2298 | 0.9754 | 0.4693 | 0.2363 | 0.1288 | 0.9719 | 0.1770 | 0.0836 | 0.9826 | 0.2066 | 0.0954 | 0.9788 |
| BPNet [75] | 0.3573 | 0.2077 | 0.9482 | 0.2392 | 0.1120 | 0.9744 | 0.1757 | 0.0793 | 0.9829 | 0.2220 | 0.1040 | 0.9765 |
| DepthPrompting [19] | 0.3583 | 0.2067 | 0.9101 | 0.2195 | 0.1006 | 0.9733 | 0.2101 | 0.1008 | 0.9743 | 0.2335 | 0.1191 | 0.9686 |
| Ours | **0.2099** | **0.1075** | **0.9752** | **0.1657** | **0.0794** | **0.9849** | **0.1473** | **0.0669** | **0.9885** | **0.1632** | **0.0745** | **0.9860** |

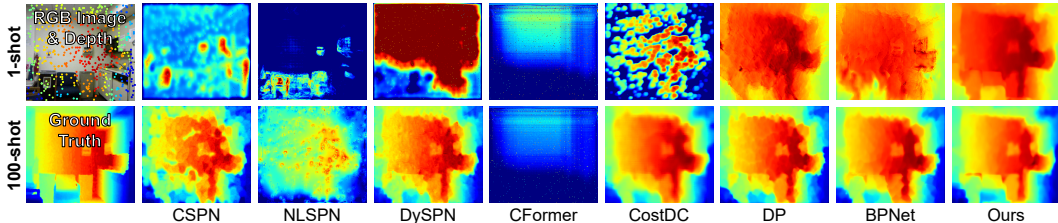

Figure 3: Results of 1-/100-shot on NYU. (CFormer: CompletionFormer, DP: DepthPrompting).

Table 2: Ablation of hyperbolic operations on zero-/few-shot performance for NYU and KITTI.

| | NYU w/o Training | KITTI w/o Training | NYU 1-shot | NYU 10-shot | NYU 100-shot | KITTI 1-shot | KITTI 10-shot | KITTI 100-shot |
|---|---|---|---|---|---|---|---|---|
| Euclidean | 13.889 / 11.507 | 77.642 / 63.109 | 0.217 / 0.112 | 0.172 / 0.081 | 0.149 / 0.069 | 1.745 / 0.578 | 1.397 / 0.417 | 1.291 / 0.342 |
| Hyperbolic | **0.323 / 0.246** | **4.061 / 1.974** | **0.210 / 0.108** | **0.166 / 0.079** | **0.147 / 0.067** | **1.684 / 0.522** | **1.385 / 0.407** | **1.224 / 0.339** |

only a single pair of an image and its corresponding dense depth, our model demonstrates a substantial performance advantage over the comparison models. This underscores the effectiveness of using the foundation model that does not require any additional learning for new depth representations of unseen data. The models with a large number of parameters to learn, such as CompletionFormer [80] (83.6M), often struggle to optimize with limited datasets. Since the depth prompting module in [19] requires training from scratch, it encounters difficulties in the adaptation to new sensors.

Table 4: Result of few-shot learning without dense GT depth. (RMSE/MAE)

| KITTIDC | | 1-shot | 10-shot | 100-shot |
|---|---|---|---|---|
| BPNet (8-Line) | | 11.64 / 3.19 | 4.00 / 1.62 | 3.28 / 1.36 |
| DepthPrompting (8-Line) | | 8.15 / 5.67 | 6.77 / 3.75 | 5.05 / 2.36 |
| Ours (8-Line) | | **4.34 / 1.77** | **3.32 / 1.33** | **2.89 / 1.12** |
| BPNet (32-Line) | | 4.76 / 1.54 | 2.56 / 0.82 | 2.08 / 0.72 |
| DepthPrompting (32-Line) | | 3.90 / 1.63 | 2.92 / 1.25 | 2.40 / 0.87 |
| Ours (32-Line) | | **2.01 / 0.66** | **1.92 / 0.61** | **1.89 / 0.64** |

***Few-shot learning without dense GT.*** Training without dense GT depths is a more practical scenario because obtaining high-quality and metric-scale depth is difficult, particularly in outdoor datasets. To validate the applicability, we train our model in a self-supervised manner without a dense GT depth. Specifically, the input LiDAR is sampled at 8-line and 32-line, while the supervision is provided by 64-Line LiDAR. This approach enables our model to adapt to sparser LiDAR inputs without the need for dense supervision. As shown in Tab.4, these results highlight our model's robustness and superior adaptation capabilities over BPNet [76] and DepthPromtping [19], which are the 2nd/3rd best in Tab.1 and Tab.3,

## 5.3 Ablation Study

***Probe for hyperbolic embedding.*** We assess the efficacy of hyperbolic embedding and curvature generation, focusing on their performance in zero-shot settings. In Tab.2, the hyperbolic method yields promising results, whereas the Euclidean approach fails. The performance gap implies that hyperbolic space offers discriminative features which guide the sparse depth propagation well. While the influence of initial parameter settings cannot be overlooked, the potential for rapid adaptation can be enhanced through well-devised initialization methods, which are in line with principles from meta-learning strategies [82, 83]. we conduct additional experiments under the few-shot regime. The results, presented in Table.F, show a noticeable improvement when using hyperbolic space, with a performance gain of 5% on average, compared to Euclidean space. This validates the effectiveness of hyperbolic geometry in depth completion tasks, especially when dealing with limited data samples.

Additionally, the analysis of the multi-curvature approach for the refinement process (Tab.5) reveals that the curvature values for multi-size affinity maps in-

Table 5: Averaged curvature values.

| Multi-curvatrue (Eq.11) | | NYU | KITTIDC |
|---|---|---|---|
| $\kappa_k, (k = \{3, 5, 7\})$ | | 0.48, 0.23, 0.07 | 0.41,0.14,0.17 |

Table 3: Quantitative results on KITTI DC.

| | 1-Shot | | | 10-Shot | | | 100-Shot | | | 1-Sequence Training | | |
|---|---|---|---|---|---|---|---|---|---|---|---|---|
| | RMSE | MAE | DELTA1 | RMSE | MAE | DELTA1 | RMSE | MAE | DELTA1 | RMSE | MAE | DELTA1 |
| CSPN [25] | 9.2621 | 3.5736 | 0.9063 | 2.0061 | 0.7962 | 0.9758 | 1.4668 | 0.5018 | 0.9850 | 2.6406 | 0.8227 | 0.9679 |
| S2D [23] | 8.8701 | 5.6307 | 0.4222 | 5.0228 | 3.1807 | 0.6319 | 4.2582 | 2.6475 | 0.7030 | 4.8136 | 2.5358 | 0.8383 |
| NLSPN [78] | 7.3135 | 4.7084 | 0.5036 | 4.0327 | 2.2361 | 0.8662 | 2.4801 | 1.1862 | 0.9348 | 4.0535 | 1.7707 | 0.8787 |
| DySPN [27] | 2.6094 | 0.9082 | 0.9545 | 2.2863 | 0.8920 | 0.9487 | 1.8568 | 0.6437 | 0.9777 | 2.8369 | 0.8149 | 0.9692 |
| CompletionFormer [80] | 4.6990 | 2.4002 | 0.8224 | 3.1760 | 1.4930 | 0.9302 | 2.6263 | 1.3504 | 0.8993 | 4.5320 | 1.9842 | 0.8214 |
| BPNet [75] | 5.3724 | 1.0988 | 0.9690 | 1.8965 | 0.5317 | 0.9822 | 1.3126 | 0.3734 | 0.9915 | 2.1554 | 0.6241 | 0.9816 |
| DepthPrompting [19] | 2.9561 | 1.1657 | 0.9270 | 2.4129 | 1.1463 | 0.8943 | 1.7982 | 0.6021 | 0.9808 | 2.9616 | 0.9655 | 0.9587 |
| Ours | **1.6840** | **0.5217** | **0.9826** | **1.3850** | **0.4073** | **0.9903** | **1.2238** | **0.3386** | **0.9927** | **1.8378** | **0.5406** | **0.9824** |

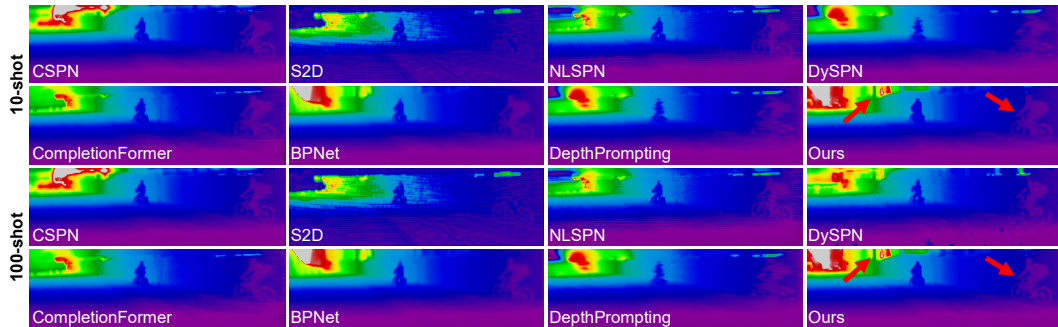

Figure 4: Qualitative results of 10-/100-shot on KITTI DC dataset.

crease with the kernel size. This trend suggests that information from more distant regions tends to prefer lower curvature. This observation supports the hypothesis that regions closer to the target require a more distinct hyperbolic space to effectively prevent bleeding errors [22]. This insight emphasizes the importance of strategic curvature adaptation for better universal depth completion.

Table 6: Ablation study on KITTI DC (RMSE / MAE ± its standard deviation).

| | 1-shot | 10-shot |
|---|---|---|
| w/o Eq.3 | 2.17±0.99 / 0.73±0.45 | 1.39±0.06 / 0.42±0.04 |
| w/o Eq.5 | 1.85±0.32 / 0.63±0.12 | 1.40±0.07 / 0.43±0.05 |
| w/o Eq.6 | 1.84±0.37 / 0.60±0.16 | 1.39±0.04 / 0.41±0.03 |
| w/o Eq.2 | 2.31±1.66 / 0.79±0.72 | 1.40±0.03 / 0.41±0.03 |
| Ours | 1.68±0.07 / 0.52±0.03 | 1.38±0.03 / 0.41±0.02 |

***Component ablation study.*** We conduct an ablation study on each component of our model as shown in Tab.6. The results reveal that removing feature fusion significantly reduces performance, particularly in 1-shot scenarios. On the other hand, the hyperbolic method shows strong adaptability with minimal data. Moreover, the fine-tuning strategy applied to the foundation model seems to be essential for the adaption to new environments, considering inherent discrepancies between relative and metric depth.

Table 7: Comparison of various depth foundation models (RMSE / MAE).

| | 1-shot (NYU) | 10-shot (NYU) | 100-shot (NYU) | 1-shot (KITTI DC) | 10-shot (KITTI DC) | 100-shot (KITTI DC) |
|---|---|---|---|---|---|---|
| DepthAnything [47] | 0.2325 / 0.1211 | 0.2022 / 0.0976 | 0.1885 / 0.0843 | 1.9900 / 0.8004 | 1.4860 / 0.4339 | 1.5703 / 0.5077 |
| UniDepth [84] | **0.2149 / 0.1099** | 0.2020 / 0.0961 | 0.1739 / 0.0768 | **1.7102** / 0.6219 | 1.4422 / 0.4340 | 1.4041 / 0.4417 |
| MiDaS [46] | 0.2169 / 0.1159 | **0.1627 / 0.0809** | **0.1455 / 0.0660** | 1.7957 / **0.5651** | **0.13930 / 0.4158** | **1.2347 / 0.3535** |

Table 8: Computational cost of Models.

| Model | Total Param. | Learnable Param. | Inference Time(s) | GPU Memory(MiB) |
|---|---|---|---|---|
| BPNet [75] | 89.874M | 89.874M | 0.072 | 4792 |
| LRRU [85] | 20.843M | 20.843M | 0.038 | 3650 |
| CompletionFormer [80] | 83.574M | 83.574M | 0.060 | 4206 |
| Ours_MiDaS [46] | 21.279M | 4.685M | 0.056 | 5980 |
| Ours_DepthAnything [47] | 24.731M | 2.981M | 0.035 | 4729 |
| Ours_UniDepth [84] | 238.607M | 5.852M | 0.116 | 5829 |

***Foundation model variations.*** To evaluate the versatility of our method with various foundational models, we replace our primary backbone [46] with concurrent works, DepthAnything [47] and UniDepth [84], which are foundation models for relative and metric depth estimation, respectively. Since these models are based on vision transformer (ViT) [86], differing from the convolutional version of MiDaS, we compare them without the intermediate feature fusion (Eq.3). As shown in Tab.7, while these backbones exhibit comparable performance, MiDaS [46] is more suitable for 10-shot and 100-shot scenarios. We claim that the local inductive bias of convolutions operates more flexibly in depth completion tasks, effectively propagating local information. This observation aligns

with the most state-of-the-art methods using convolutional encoder-decoder architectures with SPN refinement, as opposed to ViT-based architectures [47, 87, 84] for depth foundation models.

***Probe for computational costs of depth foundation model.*** Depth foundation models are typically large and computationally expensive due to training on extensive datasets. However, recent models offer various variants, allowing flexibility in computational demands. We conduct ablations on multiple models and observe comparable performance across them. As shown in Tab.8, MiDaS [46] and Depth Anything [47] have significantly fewer parameters than other depth completion models, suggesting that leveraging a pre-trained foundation model's knowledge does not necessarily entail high computational costs. Note that we use the publicly available official codes for MiDaS (v2.1 Small), Depth Anything v1 (ViT-S), and UniDepth (ViT-L).

Table 9: Experiment on advanced methods. To explore our method under various configurations, we develop four variants by adjusting the number of channels, similar to LRRU [85].

| Model | NYU 1-shot RMSE | MAE | DELTA1 | NYU 10-shot RMSE | MAE | DELTA1 | NYU 100-shot RMSE | MAE | DELTA1 | KITTI 1-shot RMSE | MAE | DELTA1 | KITTI 10-shot RMSE | MAE | DELTA1 | KITTI 100-shot RMSE | MAE | DELTA1 |
|---|---|---|---|---|---|---|---|---|---|---|---|---|---|---|---|---|---|---|
| LRRU_Mini | 0.704 | 0.505 | 0.738 | 0.989 | 0.677 | 0.642 | 0.551 | 0.392 | 0.797 | 6.719 | 3.068 | 0.792 | 5.608 | 2.787 | 0.811 | 3.576 | 2.020 | 0.841 |
| LRRU_Tiny | 0.842 | 0.633 | 0.574 | 0.771 | 0.549 | 0.707 | 0.565 | 0.373 | 0.836 | 7.961 | 4.049 | 0.698 | 6.253 | 3.022 | 0.788 | 4.201 | 2.394 | 0.796 |
| LRRU_Small | 0.589 | 0.388 | 0.826 | 0.404 | 0.246 | 0.919 | 0.442 | 0.306 | 0.887 | 16.162 | 8.008 | 0.516 | 5.930 | 2.905 | 0.800 | 5.934 | 3.602 | 0.612 |
| LRRU_Base | 0.447 | 0.278 | 0.899 | 0.424 | 0.252 | 0.922 | 0.316 | 0.189 | 0.949 | 14.889 | 7.454 | 0.526 | 13.078 | 6.904 | 0.587 | 9.736 | 6.090 | 0.420 |
| DFU | 0.754 | 0.589 | 0.637 | 0.590 | 0.464 | 0.719 | 0.467 | 0.344 | 0.868 | 3.652 | 2.020 | 0.853 | 1.889 | 0.966 | 0.973 | 1.808 | 0.897 | 0.986 |
| OGNI-DC | 0.365 | 0.200 | 0.921 | 0.312 | 0.160 | 0.957 | 0.207 | 0.095 | 0.974 | 2.618 | 0.816 | 0.962 | 1.516 | 0.421 | 0.985 | 1.514 | 0.430 | 0.984 |
| Ours_Mini | 0.215 | 0.116 | 0.976 | 0.161 | 0.079 | 0.986 | 0.148 | 0.071 | 0.988 | 2.051 | 0.631 | 0.978 | 1.355 | 0.405 | 0.991 | 1.252 | 0.397 | 0.992 |
| Ours_Tiny | 0.243 | 0.131 | 0.969 | 0.186 | 0.088 | 0.982 | 0.151 | 0.068 | 0.988 | 2.002 | 0.725 | 0.951 | 1.457 | 0.448 | 0.987 | 1.251 | 0.353 | 0.991 |
| **Ours** | 0.210 | 0.108 | 0.975 | 0.166 | 0.079 | 0.985 | 0.147 | 0.067 | 0.988 | 1.684 | 0.522 | 0.983 | 1.385 | 0.407 | 0.990 | 1.224 | 0.339 | 0.993 |
| Ours_Small | 0.255 | 0.136 | 0.968 | 0.181 | 0.089 | 0.983 | 0.149 | 0.067 | 0.988 | 1.865 | 0.590 | 0.975 | 1.465 | 0.436 | 0.988 | 1.283 | 0.388 | 0.991 |
| Ours_Base | 0.247 | 0.138 | 0.969 | 0.190 | 0.093 | 0.983 | 0.148 | 0.066 | 0.988 | 1.716 | 0.607 | 0.979 | 1.423 | 0.428 | 0.988 | 1.246 | 0.345 | 0.992 |

Table 10: Experiment on SUN RGB-D dataset.

| Model | 1-shot RMSE | MAE | DELTA1 | 10-shot RMSE | MAE | DELTA1 | 100-shot RMSE | MAE | DELTA1 |
|---|---|---|---|---|---|---|---|---|---|
| BPNet | - | - | - | 0.497 | 0.244 | 0.870 | 0.342 | 0.164 | 0.900 |
| DP | 0.706 | 0.534 | 0.537 | 0.683 | 0.512 | 0.558 | 0.700 | 0.527 | 0.545 |
| LRRU | 0.912 | 0.743 | 0.347 | 0.507 | 0.300 | 0.785 | 0.476 | 0.313 | 0.779 |
| DFU | - | - | - | 0.890 | 0.696 | 0.314 | 0.552 | 0.444 | 0.438 |
| OGNI-DC | - | - | - | 0.466 | 0.270 | 0.817 | 0.382 | 0.188 | 0.881 |
| Ours | **0.529** | **0.285** | **0.830** | **0.418** | **0.188** | **0.895** | **0.345** | **0.166** | **0.901** |

***Additional experiments on recent SoTA methods and other sensor.*** In Tab.9, we compare our approach with recent SoTA methods, showing its advantages across different experimental setups. Unlike the LRRU family [85], which performs variably across datasets due to the IP-Basic algorithm's KITTI dataset bias, our model leverages foundation model knowledge for consistent adaptation to both indoor and outdoor environments. DFU [88] and OGNIDC [89] introduce depth feature upsampling and gradient refinement, respectively. Our method, however, efficiently learns hyperbolic representations on smaller datasets, enabling faster adaptation in challenging conditions. Additionally, we evaluate our model on SUN-RGBD as shown in Tab.10, containing diverse RGB-D images from multiple sensors (Intel RealSense, Asus Xtion, Kinect V1/V2), with consistent improvements across these sensors.

## 6 Conclusion

This work starts from the new problem definition, Universal Depth Completion, to tackle the challenge of consistent depth estimation across diverse scenes and sensors. We propose a simple yet universally applicable framework that leverages the knowledge of the depth foundational model and few-shot learning capabilities using hyperbolic geometry. Through various experiments in few-/zero-shot scenarios, we validate the adaptability and generality of our method.

***Limitation & Future work.*** There are rooms for improvement. In this work, we can only use a pair of an image and corresponding sparse depth as input. For general full 3D reconstruction and novel view synthesis, our method is needed to handle input pairs with multiple viewpoints. In addition, the direct application to another modality like radar is challenging due to the noisy and highly sparse nature of the radar-derived depth information. For this, we have to devise a method to estimate an uncertainty on the noisy measurements, which will be one of interesting future works.

**Acknowledgement.** This work was supported by the National Research Foundation of Korea(NRF) grant funded by the Korea government(MSIT)(RS-2024-00338439), the Institute of Information & communications Technology Planning & Evaluation (IITP) grant funded by the Korea government (MSIT) (No.2019-0-01842, Artificial Intelligence Graduate School Program (GIST), RS-2021-II212068, Artificial Intelligence Innovation Hub), GIST-MIT Research Collaboration grant funded by the GIST in 2024, 'Project for Science and Technology Opens the Future of the Region' program through the INNOPOLIS FOUNDATION funded by Ministry of Science and ICT (Project Number: 2022-DD-UP-0312), and Local Finance Association(LOFA) grant funded by the Korea government(A Study on the Complete Survey of Advertisements Using Artificial Intelligence (AI) Technology).

# References

[1] Maximilian Jaritz, Raoul De Charette, Emilie Wirbel, Xavier Perrotton, and Fawzi Nashashibi. Sparse and dense data with cnns: Depth completion and semantic segmentation. In *2018 International Conference on 3D Vision (3DV)*, pages 52–60. IEEE, 2018. 1

[2] Ce Liu, Suryansh Kumar, Shuhang Gu, Radu Timofte, and Luc Van Gool. Single image depth prediction made better: A multivariate gaussian take. In *Proceedings of IEEE Conference on Computer Vision and Pattern Recognition (CVPR)*, 2023. 1

[3] Shuwei Shao, Zhongcai Pei, Weihai Chen, Xingming Wu, and Zhengguo Li. Nddepth: Normal-distance assisted monocular depth estimation. In *Proceedings of IEEE Conference on Computer Vision and Pattern Recognition (CVPR)*, 2023. 1

[4] Zhiqiang Yan, Kun Wang, Xiang Li, Zhenyu Zhang, Guangyu Li, Jun Li, and Jian Yang. Learning complementary correlations for depth super-resolution with incomplete data in real world. *IEEE transactions on neural networks and learning systems*, 2022. 1

[5] Fangchang Ma, Guilherme Venturelli Cavalheiro, and Sertac Karaman. Self-supervised sparse-to-dense: Self-supervised depth completion from lidar and monocular camera. In *Proceedings of IEEE International Conference on Robotics and Automation (ICRA)*, 2019. 1

[6] Kyeongha Rho, Jinsung Ha, and Youngjung Kim. Guideformer: Transformers for image guided depth completion. In *Proceedings of IEEE Conference on Computer Vision and Pattern Recognition (CVPR)*, 2022. 1

[7] Zhiqiang Yan, Xiang Li, Kun Wang, Zhenyu Zhang, Jun Li, and Jian Yang. Multi-modal masked pre-training for monocular panoramic depth completion. In *Proceedings of European Conference on Computer Vision (ECCV)*, pages 378–395. Springer, 2022. 1

[8] Zhiqiang Yan, Xiang Li, Kun Wang, Shuo Chen, Jun Li, and Jian Yang. Distortion and uncertainty aware loss for panoramic depth completion. In *Proceedings of the International Conference on Machine Learning (ICML)*, 2023. 1

[9] Kun Wang, Zhenyu Zhang, Zhiqiang Yan, Xiang Li, Baobei Xu, Jun Li, and Jian Yang. Regularizing nighttime weirdness: Efficient self-supervised monocular depth estimation in the dark. In *Proceedings of International Conference on Computer Vision (ICCV)*, 2021. 1

[10] Zhiqiang Yan, Kun Wang, Xiang Li, Zhenyu Zhang, Jun Li, and Jian Yang. Desnet: Decomposed scale-consistent network for unsupervised depth completion. In *Proceedings of the AAAI Conference on Artificial Intelligence (AAAI)*, 2023. 1

[11] Zhiqiang Yan, Yupeng Zheng, Kun Wang, Xiang Li, Zhenyu Zhang, Shuo Chen, Jun Li, and Jian Yang. Learnable differencing center for nighttime depth perception. *arXiv preprint arXiv:2306.14538*, 2023. 1

[12] Andreas Geiger, Martin Roser, and Raquel Urtasun. Efficient large-scale stereo matching. In *Asian conference on computer vision*, pages 25–38. Springer, 2010. 1

[13] Jia-Ren Chang and Yong-Sheng Chen. Pyramid stereo matching network. In *Proceedings of IEEE Conference on Computer Vision and Pattern Recognition (CVPR)*, 2018. 1

[14] Heiko Hirschmuller and Daniel Scharstein. Evaluation of cost functions for stereo matching. In *Proceedings of IEEE Conference on Computer Vision and Pattern Recognition (CVPR)*, 2007. 1

[15] Alexandre Lopes, Roberto Souza, and Helio Pedrini. A survey on rgb-d datasets. *Computer Vision and Image Understanding*, 222:103489, 2022. 1

[16] Michael Firman. Rgbd datasets: Past, present and future. In *Proceedings of the IEEE/CVF conference on computer vision and pattern recognition workshops (CVPRW)*, 2016. 1

[17] Andreas Geiger, Philip Lenz, Christoph Stiller, and Raquel Urtasun. Vision meets robotics: The kitti dataset. *The International Journal of Robotics Research (IJRR)*, 32(11):1231–1237, 2013. 2, 17

[18] Nathan Silberman, Derek Hoiem, Pushmeet Kohli, and Rob Fergus. Indoor segmentation and support inference from rgbd images. In *Proceedings of European Conference on Computer Vision (ECCV)*, 2012. 2, 7

[19] Jin-Hwi Park, Chanhwi Jeong, Junoh Lee, and Hae-Gon Jeon. Depth prompting for sensor-agnostic depth estimation. In *Proceedings of IEEE Conference on Computer Vision and Pattern Recognition (CVPR)*, 2024. 2, 3, 7, 8, 9, 18, 19, 20

[20] Joy Hsu, Jeffrey Gu, Gong Wu, Wah Chiu, and Serena Yeung. Capturing implicit hierarchical structure in 3d biomedical images with self-supervised hyperbolic representations. In *Proceedings of the Neural Information Processing Systems (NeurIPS)*, 2021. 2, 3, 5

[21] Antonio Montanaro, Diego Valsesia, and Enrico Magli. Rethinking the compositionality of point clouds through regularization in the hyperbolic space. 35:33741–33753, 2022. 2, 3, 5

[22] Jin-Hwi Park, Jaesung Choe, Inhwan Bae, and Hae-Gon Jeon. Learning affinity with hyperbolic representation for spatial propagation. In *Proceedings of the International Conference on Machine Learning (ICML)*, 2023. 2, 3, 5, 7, 9, 19

[23] Fangchang Ma and Sertac Karaman. Sparse-to-dense: Depth prediction from sparse depth samples and a single image. In *Proceedings of IEEE International Conference on Robotics and Automation (ICRA)*, 2018. 2, 7, 9

[24] Zhenyu Li, Zehui Chen, Xianming Liu, and Junjun Jiang. Depthformer: Exploiting long-range correlation and local information for accurate monocular depth estimation. *Machine Intelligence Research*, pages 1–18, 2023. 2, 4

[25] Xinjing Cheng, Peng Wang, and Ruigang Yang. Depth estimation via affinity learned with convolutional spatial propagation network. In *Proceedings of European Conference on Computer Vision (ECCV)*, 2018. 2, 7, 8, 9, 19

[26] Xinjing Cheng, Peng Wang, Chenye Guan, and Ruigang Yang. Cspn++: Learning context and resource aware convolutional spatial propagation networks for depth completion. In *Proceedings of the AAAI Conference on Artificial Intelligence (AAAI)*, 2020. 2, 6

[27] Yuankai Lin, Tao Cheng, Qi Zhong, Wending Zhou, and Hua Yang. Dynamic spatial propagation network for depth completion. *Proceedings of the AAAI Conference on Artificial Intelligence (AAAI)*, 2022. 2, 7, 8, 9, 19

[28] Mu Hu, Shuling Wang, Bin Li, Shiyu Ning, Li Fan, and Xiaojin Gong. Penet: Towards precise and efficient image guided depth completion. In *Proceedings of IEEE International Conference on Robotics and Automation (ICRA)*, 2021. 2

[29] Xin Liu, Xiaofei Shao, Bo Wang, Yali Li, and Shengjin Wang. Graphcspn: Geometry-aware depth completion via dynamic gcns. In *Proceedings of European Conference on Computer Vision (ECCV)*, 2022. 2

[30] Brent Schwarz. Mapping the world in 3d. *Nature Photonics*, 4(7):429–430, 2010. 2

[31] Zhengyou Zhang. Microsoft kinect sensor and its effect. *IEEE multimedia*, 19(2):4–10, 2012. 2

[32] Andrea Conti, Matteo Poggi, and Stefano Mattoccia. Sparsity agnostic depth completion. In *Proceedings of the IEEE/CVF Winter Conference on Applications of Computer Vision*, pages 5871–5880, 2023. 2, 18

[33] Wei Yin, Jianming Zhang, Oliver Wang, Simon Niklaus, Simon Chen, and Chunhua Shen. Towards domain-agnostic depth completion. *arXiv preprint arXiv:2207.14466*, 2022. 2

[34] Chao Jia, Yinfei Yang, Ye Xia, Yi-Ting Chen, Zarana Parekh, Hieu Pham, Quoc Le, Yun-Hsuan Sung, Zhen Li, and Tom Duerig. Scaling up visual and vision-language representation learning with noisy text supervision. In *Proceedings of the International Conference on Machine Learning (ICML)*, 2021. 2

[35] Lu Yuan, Dongdong Chen, Yi-Ling Chen, Noel Codella, Xiyang Dai, Jianfeng Gao, Houdong Hu, Xuedong Huang, Boxin Li, Chunyuan Li, et al. Florence: A new foundation model for computer vision. *arXiv preprint arXiv:2111.11432*, 2021. 2

[36] Junnan Li, Dongxu Li, Silvio Savarese, and Steven Hoi. Blip-2: Bootstrapping language-image pretraining with frozen image encoders and large language models. *arXiv preprint arXiv:2301.12597*, 2023. 2

[37] Jean-Baptiste Alayrac, Jeff Donahue, Pauline Luc, Antoine Miech, Iain Barr, Yana Hasson, Karel Lenc, Arthur Mensch, Katherine Millican, Malcolm Reynolds, Roman Ring, Eliza Rutherford, Serkan Cabi, Tengda Han, Zhitao Gong, Sina Samangooei, Marianne Monteiro, Jacob L Menick, Sebastian Borgeaud, Andy Brock, Aida Nematzadeh, Sahand Sharifzadeh, Mikoł aj Bińkowski, Ricardo Barreira, Oriol Vinyals, Andrew Zisserman, and Karén Simonyan. Flamingo: a visual language model for few-shot learning. In *Proceedings of the Neural Information Processing Systems (NeurIPS)*, 2022. 2

[38] Menglin Jia, Luming Tang, Bor-Chun Chen, Claire Cardie, Serge Belongie, Bharath Hariharan, and Ser-Nam Lim. Visual prompt tuning. In *Proceedings of European Conference on Computer Vision (ECCV)*, 2022. 3, 4

[39] Han Cai, Chuang Gan, Ligeng Zhu, and Song Han. Tinytl: Reduce memory, not parameters for efficient on-device learning. In *Proceedings of the Neural Information Processing Systems (NeurIPS)*, 2020. 3, 4

[40] Hyojin Bahng, Ali Jahanian, Swami Sankaranarayanan, and Phillip Isola. Exploring visual prompts for adapting large-scale models. *arXiv preprint arXiv:2203.17274*, 2022. 3

[41] Hantao Yao, Rui Zhang, and Changsheng Xu. Visual-language prompt tuning with knowledge-guided context optimization. In *Proceedings of IEEE Conference on Computer Vision and Pattern Recognition (CVPR)*, 2023. 3

[42] Shoufa Chen, Chongjian Ge, Zhan Tong, Jiangliu Wang, Yibing Song, Jue Wang, and Ping Luo. Adapt-former: Adapting vision transformers for scalable visual recognition. In *Proceedings of the Neural Information Processing Systems (NeurIPS)*, 2022. 3

[43] Peng Gao, Shijie Geng, Renrui Zhang, Teli Ma, Rongyao Fang, Yongfeng Zhang, Hongsheng Li, and Yu Qiao. Clip-adapter: Better vision-language models with feature adapters. *International Journal of Computer Vision*, 132(2):581–595, 2024. 3

[44] Chong Zhou, Chen Change Loy, and Bo Dai. Extract free dense labels from clip. In *Proceedings of European Conference on Computer Vision (ECCV)*, pages 696–712. Springer, 2022. 3

[45] Jaime Spencer, Chris Russell, Simon Hadfield, and Richard Bowden. Kick back & relax: Learning to reconstruct the world by watching slowtv. In *Proceedings of International Conference on Computer Vision (ICCV)*, 2023. 3

[46] Reiner Birkl, Diana Wofk, and Matthias Müller. Midas v3.1 – a model zoo for robust monocular relative depth estimation. *arXiv preprint arXiv:2307.14460*, 2023. 3, 9, 10, 17

[47] Lihe Yang, Bingyi Kang, Zilong Huang, Xiaogang Xu, Jiashi Feng, and Hengshuang Zhao. Depth anything: Unleashing the power of large-scale unlabeled data. In *Proceedings of IEEE Conference on Computer Vision and Pattern Recognition (CVPR)*, 2024. 3, 9, 10

[48] Wenliang Zhao, Yongming Rao, Zuyan Liu, Benlin Liu, Jie Zhou, and Jiwen Lu. Unleashing text-to-image diffusion models for visual perception. In *Proceedings of International Conference on Computer Vision (ICCV)*. 3

[49] Bingxin Ke, Anton Obukhov, Shengyu Huang, Nando Metzger, Rodrigo Caye Daudt, and Konrad Schindler. Repurposing diffusion-based image generators for monocular depth estimation. *arXiv preprint arXiv:2312.02145*, 2023. 3

[50] Xiao Fu, Wei Yin, Mu Hu, Kaixuan Wang, Yuexin Ma, Ping Tan, Shaojie Shen, Dahua Lin, and Xiaoxiao Long. Geowizard: Unleashing the diffusion priors for 3d geometry estimation from a single image. 2024. 3

[51] Robin Rombach, Andreas Blattmann, Dominik Lorenz, Patrick Esser, and Björn Ommer. High-resolution image synthesis with latent diffusion models. In *Proceedings of IEEE Conference on Computer Vision and Pattern Recognition (CVPR)*, 2022. 3

[52] Maximillian Nickel and Douwe Kiela. Poincaré embeddings for learning hierarchical representations. In *Proceedings of the Neural Information Processing Systems (NeurIPS)*, 2017. 3

[53] Maximillian Nickel and Douwe Kiela. Learning continuous hierarchies in the Lorentz model of hyperbolic geometry. In *Proceedings of the International Conference on Machine Learning (ICML)*, 2018. 3

[54] Ankit Dhall, Anastasia Makarova, Octavian Ganea, Dario Pavllo, Michael Greeff, and Andreas Krause. Hierarchical image classification using entailment cone embeddings. In *Proceedings of the IEEE/CVF conference on computer vision and pattern recognition workshops (CVPRW)*, 2020. 3

[55] Mina Ghadimi Atigh, Martin Keller-Ressel, and Pascal Mettes. Hyperbolic busemann learning with ideal prototypes. In *Proceedings of the Neural Information Processing Systems (NeurIPS)*, volume 34, pages 103–115, 2021. 3

[56] Shaoteng Liu, Jingjing Chen, Liangming Pan, Chong-Wah Ngo, Tat-Seng Chua, and Yu-Gang Jiang. Hyperbolic visual embedding learning for zero-shot recognition. In *Proceedings of IEEE Conference on Computer Vision and Pattern Recognition (CVPR)*, 2020. 3

[57] Karan Desai, Maximilian Nickel, Tanmay Rajpurohit, Justin Johnson, and Shanmukha Ramakrishna Vedantam. Hyperbolic image-text representations. In *Proceedings of the International Conference on Machine Learning (ICML)*, 2023. 3

[58] Teng Long, Pascal Mettes, Heng Tao Shen, and Cees G. M. Snoek. Searching for actions on the hyperbole. In *Proceedings of IEEE Conference on Computer Vision and Pattern Recognition (CVPR)*, 2020. 3

[59] Aleksandr Ermolov, Leyla Mirvakhabova, Valentin Khrulkov, Nicu Sebe, and Ivan Oseledets. Hyperbolic vision transformers: Combining improvements in metric learning. In *Proceedings of IEEE Conference on Computer Vision and Pattern Recognition (CVPR)*, 2022. 3

[60] Mina GhadimiAtigh, Julian Schoep, Erman Acar, Nanne van Noord, and Pascal Mettes. Hyperbolic image segmentation. In *Proceedings of IEEE Conference on Computer Vision and Pattern Recognition (CVPR)*, 2022. 3, 4

[61] Dídac Surís, Ruoshi Liu, and Carl Vondrick. Learning the predictability of the future. In *Proceedings of IEEE Conference on Computer Vision and Pattern Recognition (CVPR)*, 2021. 3

[62] Luca Franco, Paolo Mandica, Bharti Munjal, and Fabio Galasso. Hyperbolic self-paced learning for self-supervised skeleton-based action representations. *arXiv preprint arXiv:2303.06242*, 2023. 3

[63] Joy Hsu, Jeffrey Gu, Gong Wu, Wah Chiu, and Serena Yeung. Capturing implicit hierarchical structure in 3d biomedical images with self-supervised hyperbolic representations. In *Proceedings of the Neural Information Processing Systems (NeurIPS)*, 2021. 3

[64] Joey Bose, Ariella Smofsky, Renjie Liao, Prakash Panangaden, and Will Hamilton. Latent variable modelling with hyperbolic normalizing flows. In *Proceedings of the International Conference on Machine Learning (ICML)*, 2020. 3

[65] Emile Mathieu, Charline Le Lan, Chris J Maddison, Ryota Tomioka, and Yee Whye Teh. Continuous hierarchical representations with poincaré variational auto-encoders. In *Proceedings of the Neural Information Processing Systems (NeurIPS)*, volume 32, 2019. 3

[66] Yoshihiro Nagano, Shoichiro Yamaguchi, Yasuhiro Fujita, and Masanori Koyama. A wrapped normal distribution on hyperbolic space for gradient-based learning. In *Proceedings of the International Conference on Machine Learning (ICML)*, 2019. 3

[67] Lingxiao Li, Yi Zhang, and Shuhui Wang. The euclidean space is evil: hyperbolic attribute editing for few-shot image generation. In *Proceedings of International Conference on Computer Vision (ICCV)*, 2023. 3, 4

[68] Zhi Gao, Yuwei Wu, Yunde Jia, and Mehrtash Harandi. Curvature generation in curved spaces for few-shot learning. In *Proceedings of International Conference on Computer Vision (ICCV)*, 2021. 3, 4

[69] Valentin Khrulkov, Leyla Mirvakhabova, Evgeniya Ustinova, Ivan Oseledets, and Victor Lempitsky. Hyperbolic image embeddings. In *Proceedings of IEEE Conference on Computer Vision and Pattern Recognition (CVPR)*, 2020. 3, 4

[70] Nurendra Choudhary, Nikhil Rao, and Chandan Reddy. Hyperbolic graph neural networks at scale: A meta learning approach. In *Proceedings of the Neural Information Processing Systems (NeurIPS)*, volume 36, 2024. 3, 4, 5

[71] Sifei Liu, Shalini De Mello, Jinwei Gu, Guangyu Zhong, Ming-Hsuan Yang, and Jan Kautz. Learning affinity via spatial propagation networks. In *Proceedings of the Neural Information Processing Systems (NeurIPS)*, 2017. 3

[72] Neil Houlsby, Andrei Giurgiu, Stanislaw Jastrzebski, Bruna Morrone, Quentin De Laroussilhe, Andrea Gesmundo, Mona Attariyan, and Sylvain Gelly. Parameter-efficient transfer learning for nlp. In *Proceedings of the International Conference on Machine Learning (ICML)*, 2019. 4

[73] Shu-Lin Xu, Yifan Sun, Faen Zhang, Anqi Xu, Xiu-Shen Wei, and Yi Yang. Hyperbolic space with hierarchical margin boosts fine-grained learning from coarse labels. In *Proceedings of the Neural Information Processing Systems (NeurIPS)*, volume 36, 2024. 5

[74] Zhi Gao, Yuwei Wu, Yunde Jia, and Mehrtash Harandi. Hyperbolic feature augmentation via distribution estimation and infinite sampling on manifolds. In *Proceedings of the Neural Information Processing Systems (NeurIPS)*, 2022. 5

[75] Jie Tang, Fei-Peng Tian, Boshi An, Jian Li, and Ping Tan. Bilateral propagation network for depth completion. In *Proceedings of IEEE Conference on Computer Vision and Pattern Recognition (CVPR)*, 2024. 6, 7, 8, 9

[76] Carlo Tomasi and Roberto Manduchi. Bilateral filtering for gray and color images. In *Proceedings of International Conference on Computer Vision (ICCV)*, 1998. 6, 8

[77] David Eigen, Christian Puhrsch, and Rob Fergus. Depth map prediction from a single image using a multi-scale deep network. In *Proceedings of the Neural Information Processing Systems (NeurIPS)*, 2014. 7

[78] Jinsun Park, Kyungdon Joo, Zhe Hu, Chi-Kuei Liu, and In So Kweon. Non-local spatial propagation network for depth completion. In *Proceedings of European Conference on Computer Vision (ECCV)*, 2020. 7, 8, 9, 19

[79] Jaewon Kam, Jungeon Kim, Soongjin Kim, Jaesik Park, and Seungyong Lee. Costdcnet: Cost volume based depth completion for a single rgb-d image. In *Proceedings of European Conference on Computer Vision (ECCV)*, 2022. 7, 8

[80] Youmin Zhang, Xianda Guo, Matteo Poggi, Zheng Zhu, Guan Huang, and Stefano Mattoccia. Completionformer: Depth completion with convolutions and vision transformers. In *Proceedings of IEEE Conference on Computer Vision and Pattern Recognition (CVPR)*, 2023. 7, 8, 9, 18

[81] Jonas Uhrig, Nick Schneider, Lukas Schneider, Uwe Franke, Thomas Brox, and Andreas Geiger. Sparsity invariant cnns. In *International Conference on 3D Vision (3DV)*, 2017. 7, 18

[82] Chelsea Finn, Pieter Abbeel, and Sergey Levine. Model-agnostic meta-learning for fast adaptation of deep networks. In *Proceedings of the International Conference on Machine Learning (ICML)*, 2017. 8

[83] Chelsea Finn, Kelvin Xu, and Sergey Levine. Probabilistic model-agnostic meta-learning. In *Proceedings of the Neural Information Processing Systems (NeurIPS)*, 2018. 8

[84] Luigi Piccinelli, Yung-Hsu Yang, Christos Sakaridis, Mattia Segu, Siyuan Li, Luc Van Gool, and Fisher Yu. Unidepth: Universal monocular metric depth estimation. In *Proceedings of IEEE Conference on Computer Vision and Pattern Recognition (CVPR)*, 2024. 9, 10

[85] Yufei Wang, Bo Li, Ge Zhang, Qi Liu, Tao Gao, and Yuchao Dai. Lrru: Long-short range recurrent updating networks for depth completion. In *Proceedings of International Conference on Computer Vision (ICCV)*, 2023. 9, 10, 18

[86] Alexey Dosovitskiy, Lucas Beyer, Alexander Kolesnikov, Dirk Weissenborn, Xiaohua Zhai, Thomas Unterthiner, Mostafa Dehghani, Matthias Minderer, Georg Heigold, Sylvain Gelly, et al. An image is worth 16x16 words: Transformers for image recognition at scale. In *Proceedings of International Conference on Learning Representations (ICLR)*, 2021. 9

[87] René Ranftl, Alexey Bochkovskiy, and Vladlen Koltun. Vision transformers for dense prediction. In *Proceedings of International Conference on Computer Vision (ICCV)*, 2021. 10

[88] Yufei Wang, Ge Zhang, Shaoqian Wang, Bo Li, Qi Liu, Le Hui, and Yuchao Dai. Improving depth completion via depth feature upsampling. In *Proceedings of IEEE Conference on Computer Vision and Pattern Recognition (CVPR)*, 2024. 10

[89] Yiming Zuo and Jia Deng. Ogni-dc: Robust depth completion with optimization-guided neural iterations. *arXiv preprint arXiv:2406.11711*, 2024. 10

[90] Abraham Albert Ungar. A gyrovector space approach to hyperbolic geometry. In *Synthesis Lectures on Mathematics and Statistics*, 2008. 17

[91] Octavian Ganea, Gary Becigneul, and Thomas Hofmann. Hyperbolic neural networks. In *Proceedings of the Neural Information Processing Systems (NeurIPS)*, 2018. 17

[92] Adam Paszke, Sam Gross, Soumith Chintala, Gregory Chanan, Edward Yang, Zachary DeVito, Zeming Lin, Alban Desmaison, Luca Antiga, and Adam Lerer. Automatic differentiation in pytorch. In *Proceedings of the Neural Information Processing Systems Workshop*, 2017. 17

[93] Diederik P Kingma and Jimmy Ba. Adam: A method for stochastic optimization. *arXiv preprint arXiv:1412.6980*, 2014. 17

[94] David Eigen, Christian Puhrsch, and Rob Fergus. Depth map prediction from a single image using a multi-scale deep network. In *Proceedings of the Neural Information Processing Systems (NeurIPS)*, 2014. 17

[95] Jonas Uhrig, Nick Schneider, Lukas Schneider, Uwe Franke, Thomas Brox, and Andreas Geiger. Sparsity invariant cnns. In *International Conference on 3D Vision (3DV)*, 2017. 17

[96] Jinyoung Jun, Jae-Han Lee, and Chang-Su Kim. Masked spatial propagation network for sparsity-adaptive depth refinement. In *Proceedings of IEEE Conference on Computer Vision and Pattern Recognition (CVPR)*, 2024. 18

[97] Jinhyung Park, Yu-Jhe Li, and Kris Kitani. Flexible depth completion for sparse and varying point densities. In *Proceedings of IEEE Conference on Computer Vision and Pattern Recognition (CVPR)*, 2024. 18

[98] Haotian Wang, Meng Yang, and Nanning Zheng. G2-monodepth: A general framework of generalized depth inference from monocular rgb+ x data. *IEEE Transactions on Pattern Analysis and Machine Intelligence*, 2023. 18

[99] Haotian Wang, Meng Yang, Xinhu Zheng, and Gang Hua. Scale propagation network for generalizable depth completion. *arXiv preprint arXiv:2410.18408*, 2024. 18

[100] Vadim Ezhov, Hyoungseob Park, Zhaoyang Zhang, Rishi Upadhyay, Howard Zhang, Chethan Chinder Chandrappa, Achuta Kadambi, Yunhao Ba, Julie Dorsey, and Alex Wong. All-day depth completion. *arXiv preprint arXiv:2405.17315*, 2024. 18

[101] Yangchao Wu, Tian Yu Liu, Hyoungseob Park, Stefano Soatto, Dong Lao, and Alex Wong. Augundo: Scaling up augmentations for monocular depth completion and estimation. In *Proceedings of European Conference on Computer Vision (ECCV)*, 2024. 18

[102] Suchisrit Gangopadhyay, Xien Chen, Michael Chu, Patrick Rim, Hyoungseob Park, and Alex Wong. Uncle: Unsupervised continual learning of depth completion. *arXiv preprint arXiv:2410.18074*, 2024. 18

[103] Hyoungseob Park, Anjali Gupta, and Alex Wong. Test-time adaptation for depth completion. In *Proceedings of IEEE Conference on Computer Vision and Pattern Recognition (CVPR)*, 2024. 18

[104] Tian Yu Liu, Parth Agrawal, Allison Chen, Byung-Woo Hong, and Alex Wong. Monitored distillation for positive congruent depth completion. In *Proceedings of European Conference on Computer Vision (ECCV)*, 2022. 18

[105] Alex Wong and Stefano Soatto. Unsupervised depth completion with calibrated backprojection layers. In *Proceedings of International Conference on Computer Vision (ICCV)*, 2021. 18

# A Appendix / supplemental material

## A.1 Hyperbolic Geometry

***Revisit to Poincaré ball model.*** We revisit some definitions of a hyperbolic ball model and the details of the fundamental arithmetic operations in the hyperbolic space. The Poincaré ball model $(\mathbb{D}_\kappa^n, \mathfrak{g}^\kappa)$ with curvature $\kappa$ is defined by a manifold $\mathbb{D}_\kappa^n = \{x \in \mathbb{R}^n \mid \kappa\|x\| < 1\}$ equipped with a metric $\mathfrak{g}^\kappa$, where $\|\cdot\|$ denotes the Euclidean norm. In contrast to traditional vector spaces, hyperbolic spaces require distinct approaches for mathematical operations. Therefore, we employ the framework of Möbius gyrovector spaces, a generalization of Euclidean vector spaces adapted for hyperbolic models. Based on Möbius transformation [90], there are fundamental arithmetic operations in the hyperbolic space, such as addition ($\oplus_\kappa$) and multiplication ($\otimes_\kappa$). Furthermore, we exploit bijective mapping functions ($\exp_0^\kappa$ and $\log_0^\kappa$) between hyperbolic space and Euclidean space.

***Möbius addition.*** For a pair $(\mathbf{u}, \mathbf{v}) \in \mathbb{D}_\kappa^{\mathbf{n}}$, the Möbius addition is defined as follows:

$$\mathbf{u} \oplus_\kappa \mathbf{v} = \frac{(\mathbf{1} + \mathbf{2}\kappa\langle\mathbf{u}, \mathbf{v}\rangle + \kappa\|\mathbf{v}\|^{\mathbf{2}})\mathbf{u} + (\mathbf{1} - \kappa\|\mathbf{u}\|^{\mathbf{2}})\mathbf{v}}{\mathbf{1} + \mathbf{2}\kappa\langle\mathbf{u}, \mathbf{v}\rangle + \kappa^{\mathbf{2}}\|\mathbf{u}\|^{\mathbf{2}}\|\mathbf{v}\|^{\mathbf{2}}}, \tag{13}$$

where $\langle\cdot, \cdot\rangle$ is the Euclidean inner product.

***Möbius matrix-vector multiplication.*** For an arbitrary function $f : \mathbb{R}^n \to \mathbb{R}^m$ in the Euclidean space, the Möbius version of $f$ is a function that maps from $\mathbb{D}^n$ to $\mathbb{D}^m$ in the hyperbolic space using Equation 4. Similarly, we can derive the Möbius matrix-vector multiplication between the matrix $\mathbf{M}$ and input $\mathbf{u}$, which is defined as:

$$\mathbf{M} \otimes_\kappa \mathbf{u} = (\mathbf{1}/\sqrt{\kappa}) \tanh\left(\frac{\|\mathbf{M}\mathbf{u}\|}{\|\mathbf{u}\|} \tanh^{-\mathbf{1}}(\sqrt{\kappa}\|\mathbf{u}\|)\right) \frac{\mathbf{M}\mathbf{u}}{\|\mathbf{M}\mathbf{u}\|}. \tag{14}$$

***Hyperbolic concatenation.*** Given image feature maps $\mathcal{F}$ in Euclidean space, we pixel-wisely embed an image feature vector at a pixel $(x, y)$ (*i.e.*, $\mathbf{f}_{(x,y)} \in \mathbb{R}^{C \times 1 \times 1}$) into the hyperbolic space. Here, we utilize an exponential mapping $\mathcal{M}(\cdot) = \exp_0^\kappa(\cdot)$ on the Poincaré ball $\mathbb{D}_\kappa^C$ as a bijective function between the Euclidean space and the hyperbolic space via Poincaré curvature $\kappa$. For concatenating features in hyperbolic space, we apply the $\beta$-concatenation proposed in [91] as below:

$$\mathcal{T}^\beta(x_1, x_2, \ldots x_N) = \mathcal{M}\left((\beta_n\beta_{n_1}^{-1}v_1^T, \ldots, \beta_n\beta_{n_N}^{-1}v_N^T)^T\right). \tag{15}$$

The points $x_i$ in the Poincare ball $D_\kappa^{n_i}$ are projected back $v_i = \mathcal{M}^{-1}(x_i)$ with the scalar coefficient $\beta_n = B(\frac{n}{2}, \frac{1}{n})$, where $B$ is the Beta distribution.

## A.2 Experiment Details

***Training details.*** We utilize MiDaS [46] as a depth foundation model whose pre-trained knowledge is transferred into our universal model. Our model is implemented with public PyTorch [92], trained on a single RTX 3090Ti GPU using Adam [93] optimizer. All training is conducted in a few-shot manner, with the number of iterations ranging from 100 to 3,000, depending on the size of the training dataset, *e.g.*, 1-shot, 10-shot, and 100-shot. Note that we resize input RGB images to keep the ratio of height/width toward MiDaS. The initial learning rate was set to $5 \times 10^{-3}$ and reduced by 0.1 every 20% for total iterations. The proposed framework comprises 4.6M learnable parameters, including 41K dedicated to tuning the foundational model. To facilitate fair comparison, each experiment is repeated 10 times with the same seeds (*e.g.*, 0 to 9), and we report the average test accuracy.

***Evaluation metrics.*** We introduce a depth quality evaluation metrics, proposed in [94, 17, 95]. We compare the competitive depth completion model and ours using official evaluation metrics: RMSE, MAE, and $\delta_{1.25}^1$. Given a ground truth depth $D = \{d\}$ and the predicted depth $\hat{D} = \{\hat{d}\}$, the metrics are as follows:

- Root mean squared error (RMSE): $\sqrt{\frac{1}{|D|} \sum_{\hat{d} \in \hat{D}} |\hat{d} - d|^2}$

- Mean absolute error (MAE): $\frac{1}{|D|} \sum_{\hat{d} \in \hat{D}} |\hat{d} - d|$

- Percentage of predicted pixels where the relative error is within a threshold ($\delta^i_{1.25}$):

$$\delta_i = \frac{card\left(\left\{\hat{d} \in \hat{D} : \max\left\{\frac{\hat{d}}{d}, \frac{d}{\hat{d}}\right\} < 1.25^i\right\}\right)}{card\left(D\right)}$$

where the $card$ is the cardinality of a set. Note that a higher $\delta_i$ indicates better prediction.

***Depth completion datsets: NYUv2 and KITTIDC.*** We employ the NYU Depth V2 dataset, which consists of 464 indoor scenes captured using a Kinect sensor. Adhering to the established train/test division, we evaluate our trained model on 215 scenes (654 samples). The NYU Depth V2 dataset offers images at 320×240 resolution. We utilize center-cropped images at 304×228 resolution and randomly select 500 points to emulate sparse depth data. For 1-sequence training setup, we choose 10 sequences from the training dataset: [conference_room_0001, study_room_0004, reception_room_0002, playroom_0006, living_room_0068, kitchen_0010, classroom_0016, bedroom_0041, bathroom_0041, basement_0001b]

For outdoor environments, we utilize the KITTI DC dataset, which comprises 90K samples. Each sample includes color images and corresponding sparse depth data, captured at approximately 6% density relative to image resolution using a Velodyne HDL-64E LiDAR sensor. The images are provided at a resolution of 1216×352. The dataset is segmented into training (86K samples), validation (7K samples), and testing (1K samples) portions. Ground truth (GT) is generated by accumulating multiple LiDAR frames and removing inaccuracies, resulting in enhanced LiDAR depths of about 20% density. For 1-sequent training setup, we utilize the following 10 sequences: [2011_09_26_drive_0001_sync, 2011_09_26_drive_0017_sync, 2011_09_26_drive_0035_sync, 2011_09_26_drive_0093_sync, 2011_09_26_drive_0106_sync, 2011_09_28_drive_0034_sync, 2011_09_28_drive_0094_sync, 2011_09_28_drive_0168_sync, 2011_09_29_drive_0004_sync, , 2011_09_30_drive_0034_sync]

## A.3 Additional Experiments

Table 11: Full Dataset Training Benchmark on NYU and KITTI dataset.

| # of Learnable (M) Params. | Models | NYU | | KITTI | |
|---|---|---|---|---|---|
| | | RMSE | MAE | RMSE | MAE |
| 41.5M | Cformer_Tiny | 0.091 | 0.035 | - | - |
| 82.6M | Cformer_Small (Github) | 0.090 | 0.035 | 0.739 | 0.196 |
| 142.4M | Cformer_Base | 0.090 | 0.035 | 0.709 | 0.203 |
| 0.3M | LRRU_Mini | 0.101 | - | 0.800 | 0.219 |
| 1.3M | LRRU_Tiny | 0.096 | - | 0.762 | 0.208 |
| 5.2M | LRRU_Small | 0.093 | - | 0.741 | 0.202 |
| 21M | LRRU_Base | 0.091 | - | 0.728 | 0.198 |
| 1.2M | Ours_Tiny | 0.107 | 0.042 | 0.907 | 0.231 |
| 4.6M | **Ours** | **0.098** | **0.038** | **0.867** | **0.224** |
| 36.9M | Ours_Small | 0.095 | 0.038 | 0.824 | 0.209 |
| 63.2M | Ours_Base | 0.093 | 0.036 | **-** | **-** |

***Full dataset training benchmark on NYU and KITTI dataset.*** We report the performance of our work in the KITTI benchmark [81], which is reported in Tab.11. To analyze our method under various configurations, similar to recent SoTAs such as LRRU [85] and CompletionFormer [80], we designed four variants by adjusting the number of channels. Notably, both our method and these methods totally follow the scaling laws of deep learning models. Our variants achieve competitive results compared to the SoT methods, especially in setups with fewer labels.

We emphasize that, over the past decade, numerous depth completion papers have focused on in-domain experiments on NYU and KITTI datasets. However, there has been a recent trend towards addressing out-of-domain challenges in depth completion research [32, 96, 19, 97, 98, 99, 100, 101, 102, 103, 104, 105]. This direction aims to develop models that can handle variations in new sensor configurations, unseen environmental conditions, and training schemes without the need for dense GT. This trend is gaining attraction in top-tier conferences and journals, highlighting the importance of adaptability and generalization in depth completion models. Our research aligns with this direction and shares similar goals. Note that most of those works do not consider the KITTI benchmark, which is an in-domain experiment with a 64-Line LiDAR sensor. While we agree that top-tier papers should demonstrate a certain level of performance, we also believe that research focusing on generalization and adaptability for arbitrary sensors and environments is valuable and deserves recognition.

Table 12: Ablation study on SPNs with hyperbolic operation.

| NYU | 1-shot | | | 10-shot | | | 100-shot | | |
|---|---|---|---|---|---|---|---|---|---|
| Model | RMSE | MAE | DELTA1 | RMSE | MAE | DELTA1 | RMSE | MAE | DELTA1 |
| CSPN [25] | 1.483 | 1.213 | 0.266 | 0.470 | 0.330 | 0.839 | 0.222 | 0.106 | 0.973 |
| CSPN + Hyp | 1.188 | 0.950 | 0.398 | 0.429 | 0.271 | 0.866 | 0.186 | 0.101 | 0.982 |
| NLSPN [78] | 1.396 | 1.136 | 0.290 | 0.925 | 0.719 | 0.489 | 0.283 | 0.192 | 0.952 |
| NLSPN + Hyp | 1.338 | 1.079 | 0.328 | 0.353 | 0.208 | 0.934 | 0.211 | 0.133 | 0.978 |
| DySPN [27] | 1.499 | 1.210 | 0.283 | 0.567 | 0.422 | 0.742 | 0.243 | 0.117 | 0.972 |
| DySPN + Hyp | 1.303 | 1.044 | 0.035 | 0.428 | 0.304 | 0.807 | 0.216 | 0.103 | 0.978 |

***Advantages of hyperbolic space for calculation of the pixel affinity map.*** Most spatial propagation networks (e.g., CSPN [25], NLSPN [78], and DySPN [27]) adopt encoder-decoder structures to extract multi-scale features w.r.t. structure and photometric similarities. Then, initial seeds (i.e., sparse depth) are propagated based on affinity maps computed from the learned features in an iterative manner. Therefore, if the computed affinity map is accurate, capturing boundary information, which is the highly ambiguous region for pixel-wise prediction task, is concomitant. However, object boundary ambiguities, caused by noise or smooth intensity changes, can lead to bleeding errors [22]. To address these issues, we formulate these hierarchical relations in a continuous and differentiable manner. The hyperbolic space naturally accommodates exponentially growing hierarchies and tree-like structures, allowing robust affinity construction with low distortion.

We conducted a toy example to verify the effectiveness of hyperbolic geometry in various propagation schemes, including CSPN (Convolutional), NLSPN (Non-Local), and DySPN (Dynamic attention). Using the same backbone (ResNet-34) and loss functions (L1 and L2) across all schemes ensures a fair comparison. As shown in Tab.12, hyperbolic operations significantly improve performance in various few-shot setups. Compared to Euclidean methods, hyperbolic structures improve pixel distinction under challenging conditions.

Table 13: Ablation study on VFM (Visual Foundation Model) for various depth completion models.

| NYU | 1-shot | | | 10-shot | | | 100-shot | | |
|---|---|---|---|---|---|---|---|---|---|
| Model | RMSE | MAE | DELTA1 | RMSE | MAE | DELTA1 | RMSE | MAE | DELTA1 |
| CSPN | 1.483 | 1.206 | 0.346 | 0.317 | 0.196 | 0.711 | 0.285 | 0.131 | 0.975 |
| CSPN + "VFM" | - | - | - | 0.569 | 0.438 | 0.756 | 0.533 | 0.408 | 0.787 |
| BPNet | 0.357 | 0.208 | 0.948 | 0.239 | 0.112 | 0.974 | 0.176 | 0.079 | 0.983 |
| BPNet + "VFM" | - | - | - | - | - | - | - | - | - |
| OGNIDC | 0.365 | 0.200 | 0.921 | 0.312 | 0.160 | 0.957 | 0.207 | 0.095 | 0.974 |
| OGNIDC + "VFM" | 0.695 | 0.323 | 0.888 | 0.372 | 0.189 | 0.932 | 0.248 | 0.148 | 0.958 |
| DepthPrompting | 0.358 | 0.207 | 0.910 | 0.220 | 0.101 | 0.973 | 0.210 | 0.101 | 0.974 |
| Ours | 0.210 | 0.108 | 0.975 | 0.166 | 0.079 | 0.985 | 0.147 | 0.067 | 0.988 |

***Ablation study on VFM (Visual Foundation Model).*** We carry out additional experiments using VFM knowledge in conventional methods by replacing the sparse depth input with Eq.4 of the paper [19]. In this experiment, we found that directly applying the VFM approach, as suggested, sometimes yields unsatisfactory performance compared to the baseline, as shown in Tab.13. This underperformance can be attributed to optimization issues stemming from the fact that the sparse depth provides complete metric depth information, whereas fitting the relative-scale depth from VFM using Eq.4 of DepthPrompting [19] does not achieve this precision level. The fitting process involves solving AxB, which performs a linear fit with the available data, i.e., sparse depth. In [19], the authors employed global linear fitting to predict the depth scale using scalar values A and B, initially converting relative depth to metric scale. However, this approach often fits disproportionately to regions with rich information, leading to inaccuracies in areas with sparse depth information. Consequently, using metric sparse depth as input can cause inaccuracies, making optimization difficult and resulting in suboptimal performance. We agree that there is a significant gap in using VFM directly for depth completion. Instead of directly using relative-scale depth, we chose to leverage intermediate features to indirectly utilize foundation knowledge. This approach allows us to benefit from VFM while avoiding the direct application of relative-scale depth, thereby mitigating some of the challenges observed in this experiment.

Table 14: Varying density experiment.

| NYU #Sample 100 | 1-shot | | | 10-shot | | | 100-shot | | |
|---|---|---|---|---|---|---|---|---|---|
| Model ‖ | RMSE | MAE | DELTA1 | RMSE | MAE | DELTA1 | RMSE | MAE | DELTA1 |
| BPNet | 0.737 | 0.436 | 0.876 | 0.319 | 0.177 | 0.942 | 0.276 | 0.149 | 0.955 |
| LRRU | - | - | - | 0.512 | 0.344 | 0.849 | 0.453 | 0.184 | 0.927 |
| OGNIDC | 0.439 | 0.274 | 0.884 | 0.394 | 0.176 | 0.933 | 0.287 | 0.154 | 0.951 |
| Ours | 0.326 | 0.196 | 0.936 | 0.261 | 0.151 | 0.962 | 0.227 | 0.196 | 0.971 |
| NYU #Sample 32 | 1-shot | | | 10-shot | | | 100-shot | | |
| Model ‖ | RMSE | MAE | DELTA1 | RMSE | MAE | DELTA1 | RMSE | MAE | DELTA1 |
| BPNet | 0.676 | 0.486 | 0.763 | 0.492 | 0.326 | 0.851 | 0.403 | 0.258 | 0.888 |
| LRRU | - | - | - | 0.735 | 0.547 | 0.688 | 0.678 | 0.496 | 0.719 |
| Ours | 0.486 | 0.325 | 0.852 | 0.380 | 0.244 | 0.893 | 0.312 | 0.190 | 0.935 |
| KITTI 16-Line | 1-shot | | | 10-shot | | | 100-shot | | |
| Model ‖ | RMSE | MAE | DELTA1 | RMSE | MAE | DELTA1 | RMSE | MAE | DELTA1 |
| BPNet | 3.387 | 1.203 | 0.954 | 3.063 | 1.086 | 0.964 | 2.305 | 0.800 | 0.975 |
| DFU | 4.357 | 2.139 | 0.862 | 3.935 | 1.885 | 0.911 | 2.990 | 1.428 | 0.950 |
| OGNIDC | 5.590 | 2.540 | 0.797 | 2.570 | 0.898 | 0.965 | 2.413 | 0.832 | 0.969 |
| Ours | 2.827 | 1.020 | 0.964 | 2.319 | 0.845 | 0.979 | 2.215 | 0.745 | 0.975 |
| KITTI 4-Line | 1-shot | | | 10-shot | | | 100-shot | | |
| Model ‖ | RMSE | MAE | DELTA1 | RMSE | MAE | DELTA1 | RMSE | MAE | DELTA1 |
| BPNet | 5.568 | 2.886 | 0.775 | 5.332 | 2.384 | 0.863 | 4.471 | 1.844 | 0.906 |
| DFU | - | - | - | 5.558 | 3.017 | 0.682 | 4.872 | 2.569 | 0.793 |
| Ours | 4.790 | 2.224 | 0.872 | 4.153 | 1.918 | 0.895 | 4.084 | 1.659 | 0.926 |

***Varying-density Performance.*** We simulate different LiDAR setups by varying the density of the input data. For the NYU indoor dataset, we randomly sampled 100 and 32 sparse depths, while for the KITTI outdoor dataset, we utilize 16-Line and 4-Line configurations. These experiments test the robustness and adaptability of our method in response to changes in input data quality and quantity. As shown in Tab.14, the results show that our method achieves superior performance across different sensor configurations. In contrast, most comparison methods exhibit a decline in performance when adapting to new sensor configurations, as demonstrated in the DepthPrompting [19].

